# Online Bounds for Bayesian Algorithms

**Sham M. Kakade**
Computer and Information Science Department
University of Pennsylvania

**Andrew Y. Ng**
Computer Science Department
Stanford University

## Abstract

We present a competitive analysis of Bayesian learning algorithms in the online learning setting and show that many simple Bayesian algorithms (such as Gaussian linear regression and Bayesian logistic regression) perform favorably when compared, in retrospect, to the single best model in the model class. The analysis does not assume that the Bayesian algorithms' modeling assumptions are "correct," and our bounds hold even if the data is adversarially chosen. For Gaussian linear regression (using logloss), our error bounds are comparable to the best bounds in the online learning literature, and we also provide a lower bound showing that Gaussian linear regression is optimal in a certain worst case sense. We also give bounds for some widely used maximum a posteriori (MAP) estimation algorithms, including regularized logistic regression.

## 1 Introduction

The last decade has seen significant progress in online learning algorithms that perform well even in adversarial settings (e.g. the "expert" algorithms of Cesa-Bianchi et al. (1997)). In the online learning framework, one makes minimal assumptions on the data presented to the learner, and the goal is to obtain good (relative) performance on arbitrary sequences. In statistics, this philosophy has been espoused by Dawid (1984) in the prequential approach.

We study the performance of Bayesian algorithms in this adversarial setting, in which the process generating the data is not restricted to come from the prior—data sequences may be arbitrary. Our motivation is similar to that given in the online learning literature and the MDL literature (see Grunwald, 2005) —namely, that models are often chosen to balance realism with computational tractability, and often assumptions made by the Bayesian are not truly believed to hold (e.g. i.i.d. assumptions). Our goal is to study the performance of Bayesian algorithms in the worst-case, where all modeling assumptions may be violated.

We consider the widely used class of generalized linear models—focusing on Gaussian linear regression and logistic regression—and provide relative performance bounds (comparing to the best model in our model class) when the cost function is the logloss. Though the regression problem has been studied in a competitive framework and, indeed, many ingenious algorithms have been devised for it (e.g., Foster, 1991; Vovk, 2001; Azoury and Warmuth, 2001) , our goal here is to study how the more widely used, and often simpler, Bayesian algorithms fare. Our bounds for linear regression are comparable to the best bounds in the literature (though we use the logloss as opposed to the square loss).

The competitive approach to regression started with Foster (1991), who provided competitive bounds for a variant of the ridge regression algorithm (under the square loss). Vovk (2001) presents many competitive algorithms and provides bounds for linear regression (under the square loss) with an algorithm that differs slightly from the Bayesian one. Azoury and Warmuth (2001) rederive Vovk's bound with a different analysis based on Bregman distances. Our work differs from these in that we consider Bayesian Gaussian

linear regression, while previous work typically used more complex, cleverly devised algorithms which are either variants of a MAP procedure (as in Vovk, 2001), or that involve other steps such as "clipping" predictions (as in Azoury and Warmuth, 2001). These distinctions are discussed in more detail in Section 3.1.

We should also note that when the loss function is the logloss, multiplicative weights algorithms are sometimes identical to Bayes rule with particular choices of the parameters (see Freund and Schapire, 1999). Furthermore, Bayesian algorithms have been used in some online learning settings, such as the sleeping experts setting of Freund et al. (1997) and the online boolean prediction setting of Cesa-Bianchi et al. (1998). Ng and Jordan (2001) also analyzed an online Bayesian algorithm but assumed that the data generation process was not too different from the model prior. To our knowledge, there have been no studies of Bayesian generalized linear models in an adversarial online learning setting (though many variants have been considered as discussed above).

We also examine maximum a posteriori (MAP) algorithms for both Gaussian linear regression (i.e., ridge regression) and for (regularized) logistic regression. These algorithms are often used in practice, particularly in logistic regression where Bayesian model averaging is computationally expensive, but the MAP algorithm requires only solving a convex problem. As expected, MAP algorithms are somewhat less competitive than full Bayesian model averaging, though not unreasonably so.

## 2 Bayesian Model Averaging

We now consider the Bayesian model averaging (BMA) algorithm and give a bound on its worst-case online loss. We start with some preliminaries. Let $x \in \mathbb{R}^n$ denote the inputs of a learning problem and $y \in \mathbb{R}$ the outputs. Consider a model from the generalized linear model family (see McCullagh and Nelder, 1989), that can be written $p(y|x, \theta) = p(y|\theta^T x)$, where $\theta \in \mathbb{R}^n$ are the parameters of our model ($\theta^T$ denotes the transpose of $\theta$). Note that the predicted distribution of $y$ depends only on $\theta^T x$, which is linear in $\theta$. For example, in the case of Gaussian linear regression, we have

$$p(y|x, \theta) = \frac{1}{\sqrt{2\pi\sigma^2}} \exp\left(\frac{-(\theta^T x - y)^2}{2\sigma^2}\right), \tag{1}$$

where $\sigma^2$ is a fixed, *known* constant that is not a parameter of our model. In logistic regression, we would have

$$\log p(y|x, \theta) = y \log \frac{1}{1 + \exp(-\theta^T x)} + (1 - y) \log\left(1 - \frac{1}{1 + \exp(-\theta^T x)}\right), \tag{2}$$

where we assume $y \in \{0, 1\}$.

Let $S = \{(x^{(1)}, y^{(1)}), (x^{(2)}, y^{(2)}), \ldots, (x^{(T)}, y^{(T)})\}$ be an arbitrary sequence of examples, possibly chosen by an adversary. We also use $S_t$ to denote the subsequence consisting of only the first $t$ examples. We assume throughout this paper that $||x^{(t)}|| \leq 1$ (where $|| \cdot ||$ denotes the $L_2$ norm).

Assume that we are going to use a Bayesian algorithm to make our online predictions. Specifically, assume that we have a Gaussian prior on the parameters:

$$p(\theta) = \mathcal{N}(\theta; \vec{0}, \nu^2 I_n),$$

where $I_n$ is the $n$-by-$n$ identity matrix, $\mathcal{N}(\cdot; \mu, \Sigma)$ is the Gaussian density with mean $\mu$ and covariance $\Sigma$, and $\nu^2 > 0$ is some fixed constant governing the prior variance. Also, let

$$p_t(\theta) = p(\theta|S_t) = \frac{\left(\prod_{i=1}^t p(y^{(i)}|x^{(i)}, \theta)\right) p(\theta)}{\int_\theta \left(\prod_{i=1}^t p(y^{(i)}|x^{(i)}, \theta)\right) p(\theta)d\theta}$$

be the posterior distribution over $\theta$ given the first $t$ training examples. We have that $p_0(\theta) = p(\theta)$ is just the prior distribution.

On iteration $t$, we are given the input $x^{(t)}$, and our algorithm makes a prediction using the posterior distribution over the outputs:

$$p(y|x^{(t)}, S_{t-1}) = \int_\theta p(y|x^{(t)}, \theta)p(\theta|S_{t-1})d\theta.$$

We are then given the true label $y^{(t)}$, and we suffer logloss $-\log p(y^{(t)}|x^{(t)}, S_{t-1})$. We define the cumulative loss of the BMA algorithm after $T$ rounds to be

$$L_{\mathrm{BMA}}(S) = \sum_{t=1}^{T} -\log p(y^{(t)}|x^{(t)}, S_{t-1}).$$

Importantly, note that even though the algorithm we consider is a Bayesian one, our theoretical results do *not* assume that the data comes from any particular probabilistic model. In particular, the data may be chosen by an adversary.

We are interested in comparing against the loss of any "expert" that uses some fixed parameters $\theta \in \mathbb{R}^n$. Define $\ell_\theta(t) = -\log p(y^{(t)}|x^{(t)}, \theta)$, and let

$$L_\theta(S) = \sum_{t=1}^{T} \ell_\theta(t) = \sum_{t=1}^{T} -\log p(y^{(t)}|x^{(t)}, \theta).$$

Sometimes, we also wish to compare against distributions over experts. Given a distribution $Q$ over $\theta$, define $\ell_Q(t) = \int_\theta -Q(\theta) \log p(y^{(t)}|x^{(t)}, \theta)d\theta$, and

$$L_Q(S) = \sum_{t=1}^{T} \ell_Q(t) = \int_\theta Q(\theta)L_\theta(S)d\theta.$$

This is the expected logloss incurred by a procedure that first samples some $\theta \sim Q$ and then uses this $\theta$ for all its predictions. Here, the expectation is over the random $\theta$, not over the sequence of examples. Note that the expectation is of the logloss, which is a different type of averaging than in BMA, which had the expectation and the log in the reverse order.

## 2.1 A Useful Variational Bound

The following lemma provides a worst case bound of the loss incurred by Bayesian algorithms and will be useful for deriving our main result in the next section. A result very similar to this (for finite model classes) is given by Freund et al. (1997). For completeness, we prove the result here in its full generality, though our proof is similar to theirs. As usual, define $KL(q||p) = \int_\theta q(\theta) \log \frac{q(\theta)}{p(\theta)}$.

**Lemma 2.1:** *Let $Q$ be any distribution over $\theta$. Then for all sequences $S$*
$$L_{\mathrm{BMA}}(S) \leq L_Q(S) + KL(Q||p_0).$$

**Proof:** Let $Y = \{y^{(1)}, \ldots, y^{(T)}\}$ and $X = \{x^{(1)}, \ldots, x^{(T)}\}$. The chain rule of conditional probabilities implies that $L_{\mathrm{BMA}}(S) = -\log p(Y|X)$ and $L_\theta(S) = -\log p(Y|X, \theta)$. So

$$
\begin{aligned}
L_{\mathrm{BMA}}(S) - L_Q(S) &= -\log p(Y|X) + \int_\theta Q(\theta) \log p(Y|X, \theta)d\theta \\
&= \int_\theta Q(\theta) \log \frac{p(Y|X, \theta)}{p(Y|X)}d\theta
\end{aligned}
$$

By Bayes rule, we have that $p_T(\theta) = \frac{p(Y|X,\theta)p_0(\theta)}{p(Y|X)}$. Continuing,

$$
\begin{aligned}
&= \int_\theta Q(\theta) \log \frac{p_T(\theta)}{p_0(\theta)}d\theta \\
&= \int_\theta Q(\theta) \log \frac{Q(\theta)}{p_0(\theta)}d\theta - \int_\theta Q(\theta) \log \frac{Q(\theta)}{p_T(\theta)}d\theta \\
&= KL(Q||p_0) - KL(Q||p_T).
\end{aligned}
$$

Together with the fact that $KL(Q||p_T) \geq 0$, this proves the lemma. $\qquad\square$

## 2.2 An Upper Bound for Generalized Linear Models

For the theorem that we shortly present, we need one new definition. Let $f_y(z) = -\log p(y|\theta^T x = z)$. Thus, $f_{y^{(t)}}(\theta^T x^{(t)}) = \ell_\theta(t)$. Note that for linear regression (as defined in Equation 1), we have that for all $y$

$$|f_y''(z)| = \frac{1}{\sigma^2} \qquad (3)$$

and for logistic regression (as defined in Equation 2), we have that for $y \in \{0, 1\}$

$$|f_y''(z)| \leq 1 \,.$$

**Theorem 2.2:** *Suppose $f_y(z)$ is continuously differentiable. Let $S$ be a sequence such that $||x^{(t)}|| \leq 1$ and such that for some constant c, $|f_{y^{(t)}}''(z)| \leq c$ (for all z). Then for all $\theta^*$,*

$$L_{\text{BMA}}(S) \leq L_{\theta^*}(S) + \frac{1}{2\nu^2}||\theta^*||^2 + \frac{n}{2}\log\left(1 + \frac{Tc\nu^2}{n}\right) \qquad (4)$$

The $||\theta^*||^2/2\nu^2$ term can be interpreted as a penalty term from our prior. The $\log$ term is how fast our loss could grow in comparison to the best $\theta^*$. Importantly, this extra loss is only logarithmic in $T$ in this adversarial setting.

This bound almost identical to those provided by Vovk (2001); Azoury and Warmuth (2001) and Foster (1991) for the linear regression case (under the square loss); the only difference is that in their bounds, the last term is multiplied by an upper bound on $y^{(t)}$. In contrast, we require no bound on $y^{(t)}$ in the Gaussian linear regression case due to the fact that we deal with the logloss (also recall $|f_y''(z)| = \frac{1}{\sigma^2}$ for all $y$).

**Proof:** We use Lemma 2.1 with $Q(\theta) = \mathcal{N}(\theta; \theta^*, \epsilon^2 I_n)$ being a normal distribution with mean $\theta^*$ and covariance $\epsilon^2 I_n$. Here, $\epsilon^2$ is a variational parameter that we later tune to get the tightest possible bound. Letting $\mathcal{H}(Q) = \frac{n}{2}\log 2\pi e\epsilon^2$ be the entropy of $Q$, we have

$$
\begin{aligned}
KL(Q||p_0) &= \int_\theta Q(\theta)\log\left[\frac{1}{(2\pi)^{n/2}|\nu^2 I_n|^{1/2}}\exp\left(-\frac{1}{2\nu^2}\theta^T\theta\right)\right]^{-1} d\theta - \mathcal{H}(Q)\\
&= n\log\nu + \frac{1}{2\nu^2}\int_\theta Q(\theta)\theta^T\theta d\theta - \frac{n}{2} - n\log\epsilon\\
&= n\log\nu + \frac{1}{2\nu^2}\left(||\theta^*||^2 + n\epsilon^2\right) - \frac{n}{2} - n\log\epsilon. \qquad (5)
\end{aligned}
$$

To prove the result, we also need to relate the error of $L_Q$ to that of $L_{\theta^*}$. By taking a Taylor expansion of $f_y$ (assume $y \in S$), we have that

$$f_y(z) = f_y(z^*) + f_y'(z^*)(z - z^*) + f_y''(\xi(z))\frac{(z - z^*)^2}{2},$$

for some appropriate function $\xi$. Thus, if $z$ is a random variable with mean $z^*$, we have

$$
\begin{aligned}
E_z[f_y(z)] &= f_y(z^*) + f_y'(z^*)\cdot 0 + E_z\left[f_y''(\xi(z))\frac{(z - z^*)^2}{2}\right]\\
&\leq f_y(z^*) + cE_z\left[\frac{(z - z^*)^2}{2}\right]\\
&= f_y(z^*) + \frac{c}{2}\text{Var}(z)
\end{aligned}
$$

Consider a single example $(x, y)$. We can apply the argument above with $z^* = \theta^{*T}x$, and $z = \theta^T x$, where $\theta \sim Q$. Note that $E[z] = z^*$ since $Q$ has mean $\theta^*$. Also, $\text{Var}(\theta^T x) = x^T(\epsilon^2 I_n)x = ||x||^2\epsilon^2 \leq \epsilon^2$ (because we previously assumed that $||x|| \leq 1$). Thus, we have

$$E_{\theta \sim Q}[f_y(\theta^T x)] \leq f_y(\theta^{*T}x) + \frac{c\epsilon^2}{2}$$

Since $\ell_Q(t) = E_{\theta \sim Q}[f_{y^{(t)}}(\theta^T x^{(t)})]$ and $\ell_{\theta^*}(t) = f_{y^{(t)}}(\theta^{*T} x^{(t)})$, we can sum both sides from $t = 1$ to $T$ to obtain

$$L_Q(S) \leq L_{\theta^*}(S) + \frac{Tc}{2}\epsilon^2$$

Putting this together with Lemma 2.1 and Equation 5, we find that

$$L_{\mathrm{BMA}}(S) \leq L_{\theta^*}(S) + \frac{Tc}{2}\epsilon^2 + n\log\nu + \frac{1}{2\nu^2}\left(||\theta^*||^2 + n\epsilon^2\right) - \frac{n}{2} - n\log\epsilon.$$

Finally, by choosing $\epsilon^2 = \frac{n\nu^2}{n+Tc\nu^2}$ and simplifying, Theorem 2.2 follows. $\qquad\square$

### 2.3 A Lower Bound for Gaussian Linear Regression

The following lower bound shows that, for linear regression, no other prediction scheme is better than Bayes in the worst case (when our penalty term is $||\theta^*||^2$). Here, we compare to an *arbitrary* predictive distribution $q(y|x^{(t)}, S_{t-1})$ for prediction at time $t$, which suffers an instant loss $\ell_q(t) = -\log q(y^{(t)}|x^{(t)}, S_{t-1})$. In the theorem, $\lfloor \cdot \rfloor$ denotes the floor function.

**Theorem 2.3:** *Let $L_{\theta^*}(S)$ be the loss under the Gaussian linear regression model using the parameter $\theta^*$, and let $\nu^2 = \sigma^2 = 1$. For any set of predictive distributions $q(y|x^{(t)}, S_{t-1})$, there exists an $S$ with $||x^{(t)}|| \leq 1$ such that*

$$\sum_{t=1}^{T} \ell_q(t) \geq \inf_{\theta^*}(L_{\theta^*}(S) + \frac{1}{2}||\theta^*||^2) + \frac{n}{2}\log\left(1 + \left\lfloor\frac{T}{n}\right\rfloor\right)$$

**Proof:** (sketch) If $n = 1$ and if $S$ is such that $x^{(t)} = 1$, one can show the equality:

$$L_{\mathrm{BMA}}(S) = \inf_{\theta^*}(L_{\theta^*}(S) + \frac{1}{2}||\theta^*||^2) + \frac{1}{2}\log(1 + T)$$

Let $Y = \{y^{(1)}, \ldots, y^{(T)}\}$ and $X = \{1, \ldots, 1\}$. By the chain rule of conditional probabilities, $L_{\mathrm{BMA}}(S) = -\log p(Y|X)$ (where $p$ is the Gaussian linear regression model), and $q$'s loss is $\sum_{t=1}^{T} \ell_q(t) = -\log q(Y|X)$. For any predictive distribution $q$ that differs from $p$, there must exist some sequence $S$ such that $-\log q(Y|X)$ is greater than $-\log p(Y|X)$ (since probabilities are normalized). Such a sequence proves the result for $n = 1$.

The modification for $n$ dimensions follows: $S$ is broken into $\lfloor T/n \rfloor$ subsequences where in every subsequence only one dimension has $x_k^{(t)} = 1$ (and the other dimensions are set to 0). The result follows due to the additivity of the losses on these subsequences. $\qquad\square$

## 3 MAP Estimation

We now present bounds for MAP algorithms for both Gaussian linear regression (i.e., ridge regression) and logistic regression. These algorithms use the maximum $\hat{\theta}_{t-1}$ of $p_{t-1}(\theta)$ to form their predictive distribution $p(y|x^{(t)}, \hat{\theta}_{t-1})$ at time $t$, as opposed to BMA's predictive distribution of $p(y|x^{(t)}, S_{t-1})$. As expected these bounds are weaker than BMA, though perhaps not unreasonably so.

### 3.1 The Square Loss and Ridge Regression

Before we provide the MAP bound, let us first present the form of the posteriors and the predictions for Gaussian linear regression. Define $A_t = \frac{1}{\nu^2}I_n + \frac{1}{\sigma^2}\sum_{i=1}^{t} x^{(i)}x^{(i)T}$, and $b_t = \sum_{i=1}^{t} x^{(i)}y^{(i)}$. We now have that

$$p_t(\theta) = p(\theta|S_t) = \mathcal{N}\left(\theta; \hat{\theta}_t, \hat{\Sigma}_t\right), \qquad (6)$$

where $\hat{\theta}_t = A_t^{-1}b_t$, and $\hat{\Sigma}_t = A_t^{-1}$. Also, the predictions at time $t + 1$ are given by

$$p(y^{(t+1)}|x^{(t+1)}, S_t) = \mathcal{N}\left(y^{(t+1)}; \hat{y}_{t+1}, s_{t+1}^2\right) \qquad (7)$$

where $\hat{y}_{t+1} = \hat{\theta}_t^T x^{(t+1)}$, $s_{t+1}^2 = x^{(t+1)T} \hat{\Sigma}_t x^{(t+1)} + \sigma^2$. In contrast, the prediction of a fixed expert using parameter $\theta^*$ would be

$$p(y^{(t)}|x^{(t)}, \theta^*) = \mathcal{N}\left(y^{(t)}; y_t^*, \sigma^2\right),\tag{8}$$

where $y_t^* = \theta^{*T} x^{(t)}$.

Now the BMA loss is:

$$L_{\text{BMA}}(S) = \sum_{t=1}^{T} \frac{1}{2s_t^2}(y^{(t)} - \hat{\theta}_{t-1}^T x^{(t)})^2 + \log \sqrt{2\pi s_t^2}\tag{9}$$

Importantly, note how Bayes is adaptively weighting the squared term with the inverse variances $1/s_t$ (which depend on the current observation $x^{(t)}$). The logloss of using a fixed expert $\theta^*$ is just:

$$L_{\theta^*}(S) = \sum_{t=1}^{T} \frac{1}{2\sigma^2}(y^{(t)} - \theta^{*T} x^{(t)})^2 + \log \sqrt{2\pi\sigma^2}\tag{10}$$

The MAP procedure (referred to as ridge regression) uses $p(y|x^{(t)}, \hat{\theta}_{t-1})$ which has a *fixed* variance. Hence, the MAP loss is essentially the square loss and we define it as such:

$$\widetilde{L}_{\text{MAP}}(S) = \frac{1}{2}\sum_{t=1}^{T}(y^{(t)} - \hat{\theta}_{t-1}^T x^{(t)})^2, \quad \widetilde{L}_{\theta^*}(S) = \frac{1}{2}\sum_{t=1}^{T}(y^{(t)} - \theta^{*T} x^{(t)})^2,\tag{11}$$

where $\hat{\theta}_t$ is the MAP estimate (see Equation 6).

**Corollary 3.1:** *Let $\gamma^2 = \sigma^2 + \nu^2$. For all $S$ such that $||x^{(t)}|| \leq 1$ and for all $\theta^*$, we have*

$$\widetilde{L}_{\text{MAP}}(S) \leq \frac{\gamma^2}{\sigma^2}\widetilde{L}_{\theta^*}(S) + \frac{\gamma^2}{2\nu^2}||\theta^*||^2 + \frac{\gamma^2 n}{2}\log\left(1 + \frac{T\nu^2}{\sigma^2 n}\right)$$

**Proof:** Using Equations (9,10) and Theorem 2.2, we have

$$\sum_{t=1}^{T} \frac{1}{2s_t^2}(y^{(t)} - \hat{\theta}_{t-1}^T x^{(t)})^2 \quad \leq \quad \sum_{t=1}^{T} \frac{1}{2\sigma^2}(y^{(t)} - \theta^{*T} x^{(t)})^2 + \frac{1}{2\nu^2}||\theta^*||^2$$

$$+ \frac{n}{2}\log\left(1 + \frac{Tc\nu^2}{n}\right) + \sum_{t=1}^{T} \log\frac{\sqrt{2\pi\sigma^2}}{\sqrt{2\pi s_t^2}}$$

Equations (6, 7) imply that $\sigma^2 \leq s_t^2 \leq \sigma^2 + \nu^2$. Using this, the result follows by noting that the last term is negative and by multiplying both sides of the equation by $\sigma^2 + \nu^2$. $\square$

We might have hoped that MAP were more competitive in that the leading coefficient, in front of the $\widetilde{L}_{\theta^*}(S)$ term in the bound, be 1 (similar to Theorem 2.2) rather than $\frac{\gamma^2}{\sigma^2}$. Crudely, the reason that MAP is not as effective as BMA is that MAP does not take into account the *uncertainty* in its predictions—thus the squared terms cannot be reweighted to take variance into account (compare Equations 9 and 11).

Some previous (non-Bayesian) algorithms did in fact have bounds with this coefficient being unity. Vovk (2001) provides such an algorithm, though this algorithm differs from MAP in that its predictions at time $t$ are a nonlinear function of $x^{(t)}$ (it uses $A_t$ instead of $A_{t-1}$ at time $t$). Foster (1991) provides a bound with this coefficient being 1 with more restrictive assumptions. Azoury and Warmuth (2001) also provide a bound with a coefficient of 1 by using a MAP procedure with "clipping." (Their algorithm thresholds the prediction $\hat{y}_t = \hat{\theta}_{t-1}^T x^{(t)}$ if it is larger than some upper bound. Note that we do not assume any upper bound on $y^{(t)}$.)

As the following lower bound shows, it is not possible for the MAP linear regression algorithm to have a coefficient of 1 for $\widetilde{L}_{\theta^*}(S)$, with a reasonable regret bound. A similar lower bound is in Vovk (2001), which doesn't apply to our setting where we have the additional constraint $||x^{(t)}|| \leq 1$.

**Theorem 3.2:** *Let $\gamma^2 = \sigma^2 + \nu^2$. There exists a sequence $S$ with $||x^{(t)}|| \leq 1$ such that*

$$\widetilde{L}_{\mathrm{MAP}}(S) \geq \inf_{\theta^*}(\widetilde{L}_{\theta^*}(S) + \frac{1}{2}||\theta^*||^2) + \Omega(T)$$

**Proof:** (sketch) Let $S$ be a length $T+1$ sequence, with $n = 1$, where for the first $T$ steps, $x^{(t)} = 1/\sqrt{T}$ and $y^{(t)} = 1$, and at $T+1$, $x^{(T+1)} = 1$ and $y^{(T+1)} = 0$. Here, one can show that $\inf_{\theta^*}(\widetilde{L}_{\theta^*}(S) + \frac{1}{2}||\theta^*||^2) = T/4$ and $\widetilde{L}_{\mathrm{MAP}}(S) \geq 3T/8$, and the result follows. $\square$

### 3.2 Logistic Regression

MAP estimation is often used for regularized logistic regression, since it requires only solving a convex program (while BMA has to deal with a high dimensional integral over $\theta$ that is intractable to compute exactly). Letting $\hat{\theta}_{t-1}$ be the maximum of the posterior $p_{t-1}(\theta)$, define $L_{\mathrm{MAP}}(S) = \sum_{t=1}^{T} -\log p(y^{(t)}|x^{(t)}, \hat{\theta}_{t-1})$. As with the square loss case, the bound we present is multiplicatively worse (by a factor of 4).

**Theorem 3.3:** *In the logistic regression model with $\nu \leq 0.5$, we have that for all sequences $S$ such that $||x^{(t)}|| \leq 1$ and $y^{(t)} \in \{0, 1\}$ and for all $\theta^*$*

$$L_{\mathrm{MAP}}(S) \leq 4L_{\theta^*}(S) + \frac{2}{\nu^2}||\theta^*||^2 + 2n\log\left(1 + \frac{T\nu^2}{n}\right)$$

**Proof:** (sketch) Assume $n = 1$ (the general case is analogous). The proof consists of showing that $\ell_{\hat{\theta}_{t-1}}(t) = -\log p(y^{(t)}|x^{(t)}, \hat{\theta}_{t-1}) \leq 4\ell_{\mathrm{BMA}}(t)$. Without loss of generality, assume $y^{(t)} = 1$ and $x^{(t)} \geq 0$, and for convenience, we just write $x$ instead of $x^{(t)}$. Now the BMA prediction is $\int_{\theta} p(1|\theta, x)p_{t-1}(\theta)d\theta$, and $\ell_{\mathrm{BMA}}(t)$ is the negative log of this. Note $\theta = \infty$ gives probability 1 for $y^{(t)} = 1$ (and this setting of $\theta$ minimizes the loss at time $t$).

Since we do not have a closed form solution of the posterior $p_{t-1}$, let us work with another distribution $q(\theta)$ in lieu of $p_{t-1}(\theta)$ that satisfies certain properties. Define $p_q = \int_{\theta} p(1|\theta, x)q(\theta)d\theta$, which can be viewed as the prediction using $q$ rather than the posterior. We choose $q$ to be the rectification of the Gaussian $\mathcal{N}(\theta; \hat{\theta}_{t-1}, \nu^2 I_n)$, such that there is positive probability only for $\theta \geq \hat{\theta}_{t-1}$ (and the distribution is renormalized). With this choice, we first show that the loss of $q$, $-\log p_q$, is less than or equal to $\ell_{\mathrm{BMA}}(t)$. Then we complete the proof by showing that $\ell_{\hat{\theta}_{t-1}}(t) \leq -4\log p_q$, since $-\log p_q \leq \ell_{\mathrm{BMA}}(t)$.

Consider the $q$ which maximizes $p_q$ subject to the following constraints: let $q(\theta)$ have its maximum at $\hat{\theta}_{t-1}$; let $q(\theta) = 0$ if $\theta < \hat{\theta}_{t-1}$ (intuitively, mass to the left of $\hat{\theta}_{t-1}$ is just making the $p_q$ smaller); and impose the constraint that $-(\log q(\theta))'' \geq 1/\nu^2$. We now argue that for such a $q$, $-\log p_q \leq \ell_{\mathrm{BMA}}(t)$. First note that due to the Gaussian prior $p_0$, it is straightforward to show that $-(\log p_{t-1})''(\theta) \geq \frac{1}{\nu^2}$ (the prior imposes some minimum curvature). Now if this posterior $p_{t-1}$ were rectified (with support only for $\theta \geq \hat{\theta}_{t-1}$) and renormalized, then such a modified distribution clearly satisfies the aforementioned constraints, and it has loss less than the loss of $p_{t-1}$ itself (since the rectification only increases the prediction). Hence, the maximizer, $q$, of $p_q$ subject to the constraints has loss less than that of $p_{t-1}$, i.e. $-\log p_q \leq \ell_{\mathrm{BMA}}(t)$.

We now show that such a maximal $q$ is the (renormalized) rectification of the Gaussian $\mathcal{N}(\theta; \hat{\theta}_{t-1}, \nu^2 I_n)$, such that there is positive probability only for $\theta > \hat{\theta}_{t-1}$. Assume some other $q_2$ satisfied these constraints and maximized $p_q$. It cannot be that $q_2(\hat{\theta}_{t-1}) < q(\hat{\theta}_{t-1})$,

else one can show $q_2$ would not be normalized (since with $q_2(\hat{\theta}_{t-1}) < q(\hat{\theta}_{t-1})$, the curvature constraint imposes that this $q_2$ cannot cross $q$). It also cannot be that $q_2(\hat{\theta}_{t-1}) > q(\hat{\theta}_{t-1})$. To see this, note that normalization and curvature imply that $q_2$ must cross $p_t$ only once. Now a sufficiently slight perturbation of this crossing point to the left, by shifting more mass from the left to the right side of the crossing point, would not violate the curvature constraint and would result in a new distribution with larger $p_q$, contradicting the maximality of $q_2$. Hence, we have that $q_2(\hat{\theta}_{t-1}) = q(\hat{\theta}_{t-1})$. This, along with the curvature constraint and normalization, imply that the rectified Gaussian, $q$, is the unique solution.

To complete the proof, we show $\ell_{\hat{\theta}_{t-1}}(t) = -\log p(1|x, \hat{\theta}_{t-1}) \leq -4 \log p_q$. We consider two cases, $\hat{\theta}_{t-1} < 0$ and $\hat{\theta}_{t-1} \geq 0$. We start with the case $\hat{\theta}_{t-1} < 0$. Using the boundedness of the derivative $|\partial \log p(1|x, \theta)/\partial \theta| < 1$ and that $q$ only has support for $\theta \geq \hat{\theta}_{t-1}$, we have

$$
\begin{aligned}
p_q &= \int_\theta \exp(\log p(1|x, \theta)) q(\theta) d\theta \\
&\leq \int_\theta \exp\left(\log(p(1|x, \hat{\theta}_{t-1}) + \theta - \hat{\theta}_{t-1}\right) q(\theta) d\theta \leq 1.6 p(1|x, \hat{\theta}_{t-1})
\end{aligned}
$$

where we have used that $\int_\theta \exp(\theta - \hat{\theta}_{t-1}) q(\theta) d\theta < 1.6$ (which can be verified numerically using the definition of $q$ with $\nu \leq 0.5$). Now observe that for $\hat{\theta}_{t-1} \leq 0$, we have the lower bound $-\log p(1|x, \hat{\theta}_{t-1}) \geq \log 2$. Hence, $-\log p_q \geq -\log p(1|x, \hat{\theta}_{t-1}) - \log 1.6 \geq (-\log p(1|x, \hat{\theta}_{t-1}))(1 - \log 1.6/\log 2) \geq 0.3 \ell_{\hat{\theta}_{t-1}}(t)$, which shows $\ell_{\hat{\theta}_{t-1}}(t) \leq -4 \log p_q$.

Now for the case $\hat{\theta}_{t-1} \geq 0$. Let $\sigma$ be the sigmoid function, so $p(1|x, \theta) = \sigma(\theta x)$ and $p_q = \int_\theta \sigma(x\theta) q(\theta) d\theta$. Since the sigmoid is concave for $\theta > 0$ and, for this case, $q$ only has support from positive $\theta$, we have that $p_q \leq \sigma\left(x \int_\theta \theta q(\theta) d\theta\right)$. Using the definition of $q$, we then have that $p_q \leq \sigma(x(\hat{\theta}_{t-1} + \nu)) \leq \sigma(\hat{\theta}_{t-1} + \nu)$, where the last inequality follows from $\hat{\theta}_{t-1} + \nu > 0$ and $x \leq 1$. Using properties of $\sigma$, one can show $|(\log \sigma)'(z)| < -\log \sigma(z)$ (for all $z$). Hence, for all $\theta \geq \hat{\theta}_{t-1}$, $|(\log \sigma)'(\theta)| < -\log \sigma(\theta) \leq -\log \sigma(\hat{\theta}_{t-1})$. Using this derivative condition along with the previous bound on $p_q$, we have that $-\log p_q \geq -\log \sigma(\hat{\theta}_{t-1} + \nu) \geq (-\log \sigma(\hat{\theta}_{t-1}))(1 - \nu) = \ell_{\hat{\theta}_{t-1}}(t)(1 - \nu)$, which shows that $\ell_{\hat{\theta}_{t-1}}(t) \leq -4 \log p_q$ (since $\nu \leq 0.5$). This proves the claim when $\hat{\theta}_{t-1} \geq 0$. $\qquad\square$

**Acknowledgments.** We thank Dean Foster for numerous helpful discussions. This work was supported by the Department of the Interior/DARPA under contract NBCHD030010.

# References

Azoury, K. S. and Warmuth, M. (2001). Relative loss bounds for on-line density estimation with the exponential family of distributions. *Machine Learning*, 43(3).

Cesa-Bianchi, N., Freund, Y., Haussler, D., Helmbold, D., Schapire, R., and Warmuth, M. (1997). How to use expert advice. *J. ACM*, 44.

Cesa-Bianchi, N., Helmbold, D., and Panizza, S. (1998). On Bayes methods for on-line boolean prediction. *Algorithmica*, 22.

Dawid, A. (1984). Statistical theory: The prequential approach. *J. Royal Statistical Society*.

Foster, D. P. (1991). Prediction in the worst case. *Annals of Statistics*, 19.

Freund, Y. and Schapire, R. (1999). Adaptive game playing using multiplicative weights. *Games and Economic Behavior*, 29:79–103.

Freund, Y., Schapire, R., Singer, Y., and Warmuth, M. (1997). Using and combining predictors that specialize. *In STOC*.

Grunwald, P. (2005). A tutorial introduction to the minimum description length principle.

McCullagh, P. and Nelder, J. A. (1989). *Generalized Linear Models (2nd ed.)*. Chapman and Hall.

Ng, A. Y. and Jordan, M. (2001). Convergence rates of the voting Gibbs classifier, with application to Bayesian feature selection. In *Proceedings of the 18th Int'l Conference on Machine Learning*.

Vovk, V. (2001). Competitive on-line statistics. *International Statistical Review*, 69.
